# MODELS WANTED: MUST FIT DIMENSIONS OF SLEEP AND DREAMING*

J. Allan Hobson,  Adam N. Mamelak[†]  and  Jeffrey P. Sutton[‡]
Laboratory of Neurophysiology and Department of Psychiatry
Harvard Medical School
74 Fenwood Road, Boston, MA 02115

## Abstract

During waking and sleep, the brain and mind undergo a tightly linked and precisely specified set of changes in state. At the level of neurons, this process has been modeled by variations of Volterra-Lotka equations for cyclic fluctuations of brainstem cell populations. However, neural network models based upon rapidly developing knowledge of the specific population connectivities and their differential responses to drugs have not yet been developed. Furthermore, only the most preliminary attempts have been made to model across states. Some of our own attempts to link rapid eye movement (REM) sleep neurophysiology and dream cognition using neural network approaches are summarized in this paper.

## 1  INTRODUCTION

New models are needed to test the closely linked neurophysiological and cognitive theories that are emerging from recent scientific studies of sleep and dreaming. This section describes four separate but related levels of analysis at which modeling may

---

*Based, in part, upon an invited address by J.A.H. at NIPS, Denver, Dec. 2 1991 and, in part, upon a review paper by J.P.S., A.N.M. and J.A.H. published in the *Psychiatric Annals*.

[†]Currently in the Department of Neurosurgery, University of California, San Francisco, CA 94143

[‡]Also in the Center for Biological Information Processing, Whitaker College, E25-201, Massachusetts Institute of Technology, Cambridge, MA 02139

be applied and outlines some of the desirable features of such models in terms of the burgeoning data of sleep and dream science. In the subsequent sections, we review our own preliminary efforts to develop models at some of the levels discussed.

## 1.1   THE INDIVIDUAL NEURON

Existing models or "neuromines" faithfully represent membrane properties but ignore the dynamic biochemical changes that change neural excitability over the long term. This is particularly important in the modeling of state control where the crucial neurons appear to act more like hormone pumps than like simple electrical transducers. Put succinctly, we need models that consider the biochemical or "wet" aspects of nerve cells, as well as the "dry" or electrical aspects (*cf.* McKenna *et al.*, in press).

## 1.2   NEURAL POPULATION INTERACTIONS

To mimic the changes in excitability of the modulatory neurons which control sleep and dreaming, new models are needed which incorporate both the engineering principles of oscillators and the biological principles of time-keeping. The latter principle is especially relevant in determining the dramatically variable long period time-constants that are observed within and across species. For example, we need to equip population models borrowed from field biology (McCarley and Hobson, 1975) with specialized properties of "wet" neurons as mentioned in section 1.1.

## 1.3   COGNITIVE CONSEQUENCES OF MODULATION OF NEURAL NETWORKS

To understand the state-dependent changes in cognition, such as those that distinguish waking and dreaming, a potentially fruitful approach is to mimic the known effects of neuromodulation and examine the information processing properties of neural networks. For example, if the input-output fidelity of networks can be altered by changing their mode (see Sutton *et al.*, this volume), we might be better able to understand the changes in both instantaneous associative properties and long term plasticity alterations that occur in sleep and dreaming. We might thus trap the brain-mind into revealing its rules for making moment-to-moment cross-correlations of its data and for changing the content and status of its storage in memory.

## 1.4   STATE-DEPENDENT CHANGES IN COGNITION

At the highest level of analysis, psychological data, even that obtained from the introspection of waking and dreaming subjects, need to be more creatively reduced with a view to modeling the dramatic alterations that occur with changes in brain state. As an example, consider the instability of orientation of dreaming, where times, places, persons and actions change without notice. Short of mastering the thorny problem of generating narrative text from a data base, and thus synthesizing an artificial dream, we need to formulate rules and measures of categorizing constancy and transformations (Sutton and Hobson, 1991). Such an approach is a

means of further refining the algorithms of cognition itself, an effort which is now limited to simple activation models that cannot change mode.

An important characteristic of the set of new models that are proposed is that each level informs, and is informed by, the other levels. This nested, interlocking feature is represented in figure 1. It should be noted that any erroneous assumptions made at level 1 will have effects at levels 2 and 3 and these will, in turn, impede our capacity to integrate levels 3 and 4. Level 4 models can and should thus proceed with a degree of independence from levels 1, 2 and 3. Proceeding from level 1 upward is the "bottom-up" approach, while proceeding from level 4 downward is the "top-down" approach. We like to think it might be possible to take both approaches in our work while according equal respect to each.

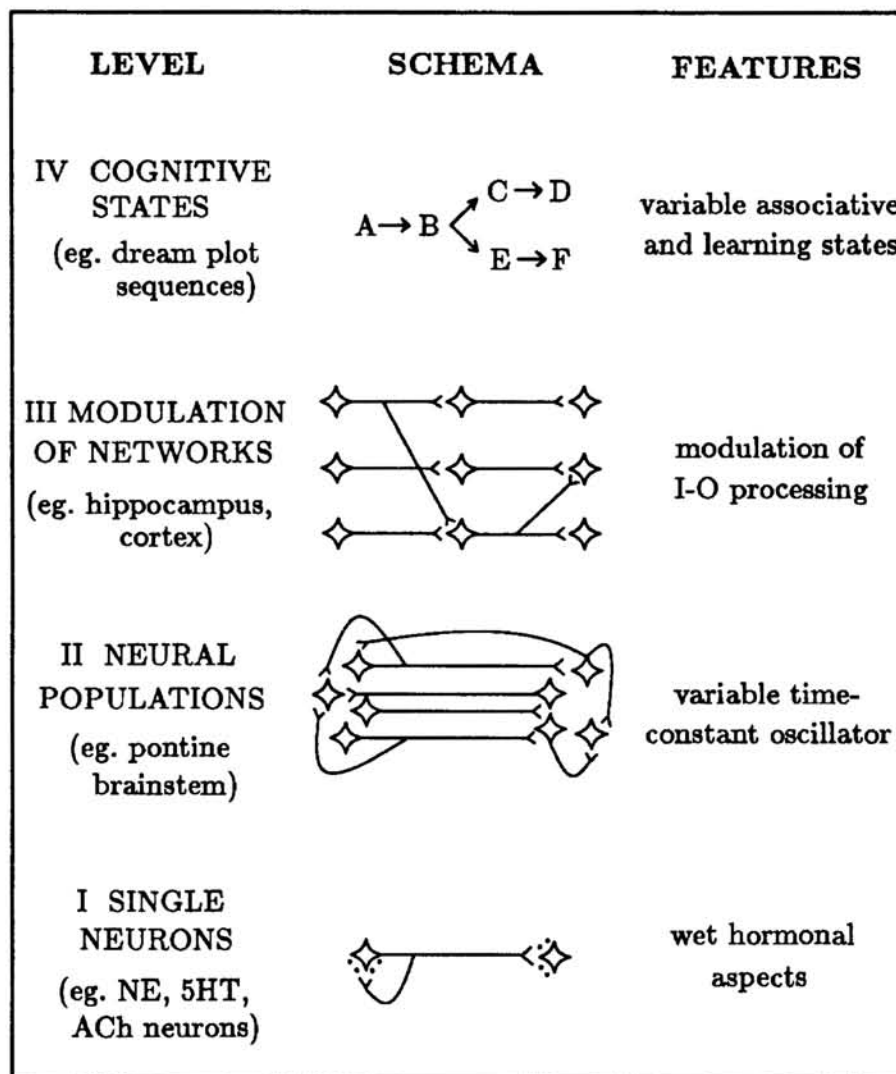

Figure 1: Four levels at which modeling innovations are needed to provide more realistic simulations of brain-mind states such as waking and dreaming. See text for discussion.

## 2   STATES OF WAKING AND SLEEPING

The states of waking and sleeping, including REM and non-REM (NREM) sleep, have characteristic behavioral, neuronal, polygraphic and psychological features that span all four levels. These properties are summarized in figures 2 and 3. Changes occurring within and between different levels are affected by the sleep-wake or circadian cycle and by the relative shifts in brain chemistry.

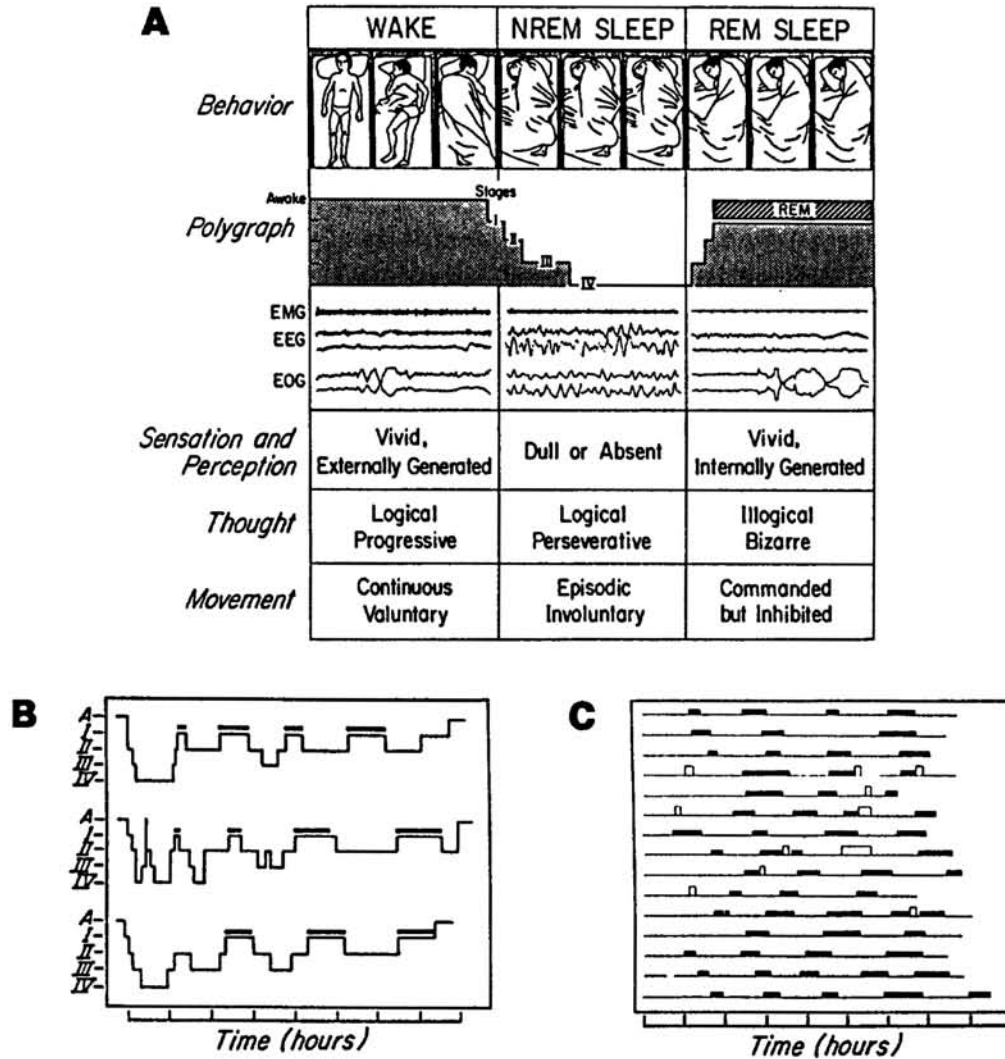

Figure 2: (a) States of waking and NREM and REM sleeping in humans. Characteristic behavioral, polygraphic and psychological features are shown for each state. (b) Ultradian sleep cycle of NREM and REM sleep shown in detailed sleep-stage graphs of 3 subjects. (c) REM sleep periodograms of 15 subjects. From Hobson and Steriade (1986), with permission.

## 2.1  CIRCADIAN RHYTHMS

The circadian cycle has been studied mathematically using oscillator and other non-linear dynamical models to capture features of sleep-wake rhythms (Moore-Ede and Czeisler, 1984; figure 2). Shorter (infradian) and longer (ultradian) rhythms, relative to the circadian rhythm, have also been examined. In general, oscillators are used to couple neural, endocrine and other pathways important in controlling a variety of functions, such as periods of rest and activity, energy conservation and thermoregulation. The oscillators can be sensitive to external cues or *zeitgebers*, such as light and daily routines, and there is a stong linkage between the circadian clock and the NREM-REM sleep oscillator.

## 2.2  RECIPROCAL INTERACTION MODEL

In the 1970s, a brainstem oscillator became identified that was central to regulating sleeping and waking. Discrete cell populations in the pons that were most active during waking, less active in NREM sleep and silent during REM sleep were found to contain the monoamines norepinephrine (NE) and serotonin (5HT). Among the many cell populations that became active during REM sleep, but were generally quiescent otherwise, were cells associated with acetylcholine (ACh) release.

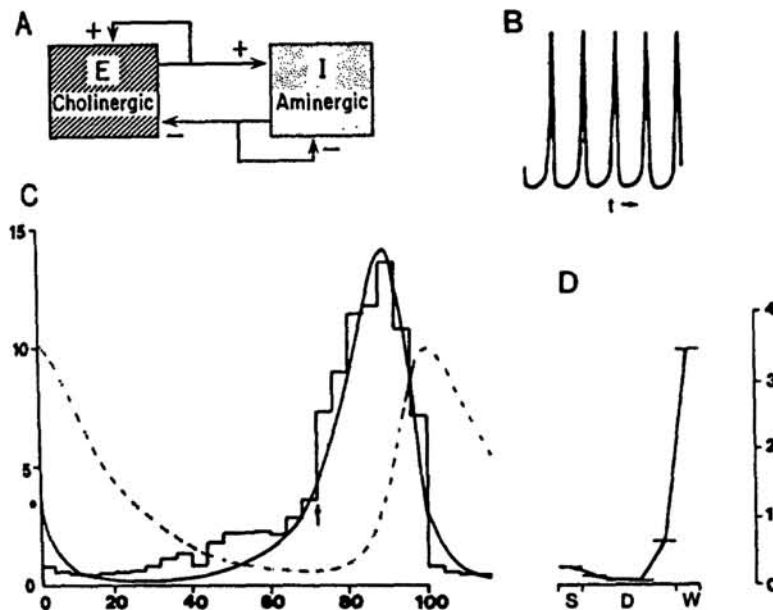

Figure 3: (a) Reciprocal interaction model of REM sleep generation showing the structural interaction between cholinergic and monoaminergic cell populations. Plus sign implies excitatory influences; minus sign implies inhibitory influences. (b) Model output of the cholinergic unit derived from Lotka-Volterra equations. (c) Histogram of the discharge rate from a cholinergic related pontine cell recorded over 12 normalized sleep-wake cycles. Model cholinergic (solid line) and monoaminergic (dotted line) outputs. (d) Noradrenergic discharge rates before (S), during (D) and following (W) a REM sleep episode. From Hobson and Steriade (1986), with permission.

By making a variety of simplifying assumptions, McCarley and Hobson (1975) were able to structurally and mathematically model the oscillations between these monoaminergic and cholinergic cell populations (figure 3). This level 2 model consists of two compartments, one being monoaminergic-inhibitory and the other cholinergic-excitatory. It is based pupon the assumptions of field biology (Volterra-Lotka) and of dry neuromines (level 3). The excitation (inhibition) originating from each compartment influences the other and also feeds back on itself. Numerous predictions generated by the model have been verified experimentally (Hobson and Steriade, 1986).

Because the neural population model shown in figure 3 uses the limited passive membrane type of neuromine discussed in the introduction, the resulting oscillator has a time-constant in the millisecond range, not even close to the real range of minutes to hours that characterize the sleep-dream cycle (figure 2). As such, the model is clearly incapable of realistically representing the long-term dynamic properties that characterize interacting neuromodulatory populations. To surmount this limitation, two modifications are possible: one is to remodel the individual neuromines equipping them with mathematics describing up and down regulation of receptors and intracellular biochemistry that results in long-term changes in synaptic efficacy (*cf.* McKenna *et al.*, in press); another is to model the longer time constants of the sleep cycle in terms of protein transport times between the two populations in brainstems of realistically varying width (*cf.* Hobson and Steriade, 1986).

## 3  NEUROCOGNITIVE ASPECTS OF WAKING, SLEEPING AND DREAMING

Since the discovery that REM sleep is correlated with dreaming, significant advances have been made in understanding both the neural and cognitive processes occurring in different states of the sleep-wake cycle. During waking, wherein the brain is in a state of relative aminergic dominance, thought content and cognition display consistency and continuity. NREM sleep mentation is typically characterized by ruminative thoughts void of perceptual vividness or emotional tone. Within this state, the aminergic and cholinergic systems are more evenly balanced than in either the wake or REM sleep states. As previously noted, REM sleep is a state associated with relative cholinergic activation. Its mental status manifestations include graphic, emotionally charged and formally bizarre images encompassing visual hallucinations and delusions.

### 3.1  ACTIVATION-SYNTHESIS MODEL

The activation-synthesis hypothesis (Hobson and McCarley, 1977) was the first account of dream mentation based on the neurophysiological state of REM sleep. It considered factors present at levels 3 and 4, according to the scheme in section 1, and attempted to bridge these two levels. In the model, cholinergic activation and reciprocal monoaminergic disinhibition of neural networks in REM sleep generated the source of dream formation. However, the details of how neural networks might actually synthesize information in the REM sleep state was not specified.

## 3.2  NEURAL NETWORK MODELS

Several neural network models have subsequently been proposed that also attempt to bridge levels 3 and 4 (for example, Crick and Mitchison, 1983). Recently, Mamelak and Hobson (1989) have suggested a neurocognitive model of dream bizarreness that extends the activation-synthesis hypothesis. In the model, the monoaminergic withdrawal in sleep relative to waking leads to a decrease in the signal-to-noise ratio in neural networks (figure 4). When this is coupled with phasic cholinergic excitation of the cortex, via brainstem ponto-geniculo-occipital (PGO) cell firing (figure 5), cognitive information becomes altered and discontinuous. A central premise of the model is that the monoamines and acetylcholine function as neuromodulators, which modify ongoing activity in networks, without actually supplying afferent input information.

Implementation of the Mamelak and Hobson model as a temporal sequencing network is described by Sutton *et al.* in this volume. Computer simulations demonstrate how changes in modulation similar to some monoaminergic and cholinergic effects can completely alter the way information is collectively sequenced within the same network. This occurs even in the absence of plastic changes in the weights connecting the artificial neurons. Incorporating plasticity, which generally involves neuromodulators such as the monoamines, is a logical next step. This would build important level 1 features into a level 3-4 model and potentially provide useful insight into some state-dependent learning operations.

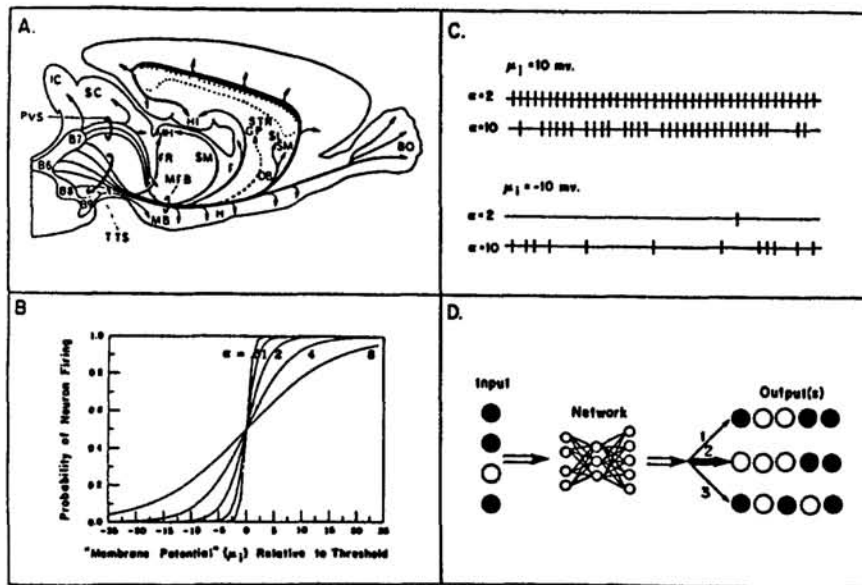

Figure 4: (a) Monoaminergic innervation of the brain is widespread. (b) Plot of the neuron firing probability as a function of the relative membrane potential for various values of monoaminergic modulation (parameterized by $\alpha$). Higher (lower) modulation is correlated with smaller (larger) $\alpha$ values. (c) Neuron firing when subjected to supra- and sub-threshold inputs of $+10$ mv and $-10$ mv, respectively, for $\alpha = 2$ and $\alpha = 10$. (d) For a given input, the repertoire of network outputs generally increases as $\alpha$ increases. From Mamelak and Hobson (1989), with permission.

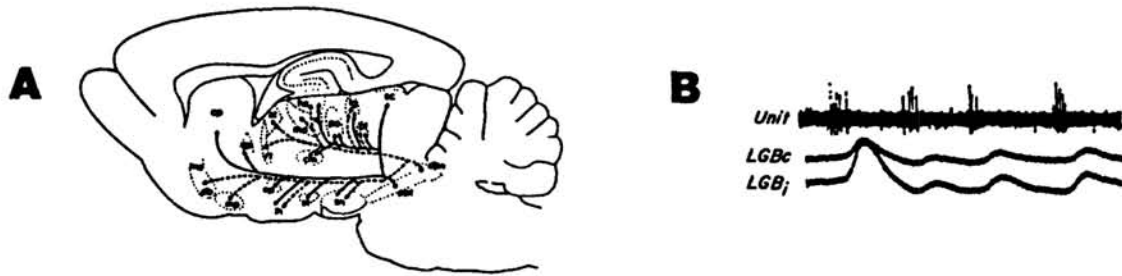

Figure 5: (a) Cholinergic input from the brainstem to the thalamus and cortex is widespread. (b) Unit recordings from PGO burst cells in the pons are correlated with PGO waves recorded in the lateral geniculate bodies (LGB) of the thalamus.

## 4    CONCLUSION

After discussing four levels at which new models are needed, we have outlined some preliminary efforts at modeling states of waking and sleeping. We suggest that this area of research is ripe for the development of integrative models of brain and mind.

**Acknowledgements**

Supported by NIH grant MH 13,923, the HMS/MMHC Research & Education Fund, the Livingston, Dupont-Warren and McDonnell-Pew Foundations, DARPA under ONR contract N00014-85-K-0124, the Sloan Foundation and Whitaker College.

**References**

Crick F, Mitchison G (1983) The function of dream sleep. *Nature* **304** 111-114.

Hobson JA, McCarley RW (1977) The brain as a dream-state generator: An activation-synthesis hypothesis of the dream process. *Am J Psych* **134** 1335-1368.

Hobson JA, Steriade M (1986) Neuronal basis of behavioral state control. In: Mountcastle VB (ed) *Handbook of Physiology - The Nervous System, Vol IV*. Bethesda: Am Physiol Soc, 701-823.

Mamelak AN, Hobson JA (1989) Dream bizarrenes as the cognitive correlate of altered neuronal behavior in REM sleep. *J Cog Neurosci* **1(3)** 201-22.

McCarley RW, Hobson JA (1975) Neuronal excitability over the sleep cycle: A structural and mathematical model. *Science* **189** 58-60.

McKenna T, Davis J, Zornetzer (eds) In press. *Single Neuron Computation*. San Diego, Academic.

Moore-Ede MC, Czeisler CA (eds) (1984) *Mathematical Models of the Circadian Sleep-Wake Cycle*. New York: Raven.

Sutton JP, Hobson (1991) Graph theoretical representation of dream content and discontinuity. *Sleep Research* **20** 164.
